# Parametric Embedding for Class Visualization

**Tomoharu Iwata, Kazumi Saito, Naonori Ueda**
NTT Communication Science Laboratories
NTT Corporation
2-4 Hikaridai Seika-Cho Soraku-gun Kyoto, 619-0237 JAPAN
{iwata,saito,ueda}@cslab.kecl.ntt.co.jp

**Sean Stromsten, Thomas L. Griffiths, Joshua B. Tenenbaum**
Department of Brain and Cognitive Sciences
Massachusetts Institute of Technology
{sean_s,gruffydd,jbt}@mit.edu

## Abstract

In this paper, we propose a new method, Parametric Embedding (PE), for visualizing the posteriors estimated over a mixture model. PE simultaneously embeds both objects and their classes in a low-dimensional space. PE takes as input a set of class posterior vectors for given data points, and tries to preserve the posterior structure in an embedding space by minimizing a sum of Kullback-Leibler divergences, under the assumption that samples are generated by a Gaussian mixture with equal covariances in the embedding space. PE has many potential uses depending on the source of the input data, providing insight into the classifier's behavior in supervised, semi-supervised and unsupervised settings. The PE algorithm has a computational advantage over conventional embedding methods based on pairwise object relations since its complexity scales with the product of the number of objects and the number of classes. We demonstrate PE by visualizing supervised categorization of web pages, semi-supervised categorization of digits, and the relations of words and latent topics found by an unsupervised algorithm, Latent Dirichlet Allocation.

## 1 Introduction

Recently there has been great interest in algorithms for constructing low-dimensional feature-space embeddings of high-dimensional data sets. These algorithms seek to capture some aspect of the data set's intrinsic structure in a low-dimensional representation that is easier to visualize or more efficient to process by other learning algorithms. Typical embedding algorithms take as input a matrix of data coordinates in a high-dimensional ambient space (e.g., PCA [5]), or a matrix of metric relations between pairs of data points (MDS [7], Isomap [6], SNE [4]). The algorithms generally attempt to map all and only nearby input points onto nearby points in the output embedding.

Here we consider a different sort of embedding problem with two sets of points $X =$

$\{\mathbf{x}_1, \ldots, \mathbf{x}_N\}$ and $C = \{c_1, \ldots, c_K\}$, which we call "objects" $(X)$ and "classes" (C). The input consists of conditional probabilities $p(c_k|\mathbf{x}_n)$ associating each object $\mathbf{x}_n$ with each class $c_k$. Many kinds of data take this form: for a classification problem, $C$ may be the set of classes, and $p(c_k|\mathbf{x}_n)$ the posterior distribution over these classes for each object $\mathbf{x}_n$; in a marketing context, $C$ might be a set of products and $p(c_k|\mathbf{x}_n)$ the probabilistic preferences of a consumer; or in language modeling, $C$ might be a set of semantic topics, and $p(c_k|\mathbf{x}_n)$ the distribution over topics for a particular document, as produced by a method like Latent Dirichlet Allocation (LDA) [1]. Typically, the number of classes is much smaller than the number of objects, $K << N$.

We seek a low-dimensional embedding of both objects and classes such that the distance between object $n$ and class $k$ is monotonically related to the probability $p(c_k|\mathbf{x}_n)$. This embedding simultaneously represents not only the relations between objects and classes, but also the relations within the set of objects and within the set of classes – each defined in terms of relations to points in the other set. That is, objects that tend to be associated with the same classes should be embedded nearby, as should classes that tend to have the same objects associated with them. Our primary goals are visualization and structure discovery, so we typically work with two- or three-dimensional embeddings.

Object-class embeddings have many potential uses, depending on the source of the input data. If $p(c_k|\mathbf{x}_n)$ represents the posterior probabilities from a supervised Bayesian classifier, an object-class embedding provides insight into the behavior of the classifier: how well separated the classes are, where the errors cluster, whether there are clusters of objects that "slip through a crack" between two classes, which objects are not well captured by any class, and which classes are intrinsically most confusable with each other. Answers to these questions could be useful for improved classifier design. The probabilities $p(c_k|\mathbf{x}_n)$ may also be the product of unsupervised or semi-supervised learning, where the classes $c_k$ represent components in a generative mixture model. Then an object-class embedding shows how well the intrinsic structure of the objects (and, in a semi-supervised setting, any given labels) accords with the clustering assumptions of the mixture model.

Our specific formulation of the embedding problem assumes that each class $c_k$ can be represented by a spherical Gaussian distribution in the embedding space, so that the embedding as a whole represents a simple Gaussian mixture model for each object $\mathbf{x}_n$. We seek an embedding that matches the posterior probabilities for each object under this Gaussian mixture model to the input probabilities $p(c_k|\mathbf{x}_n)$. Minimizing the Kullback-Leibler (KL) divergence between these two posterior distributions leads to an efficient algorithm, which we call *Parametric Embedding* (PE).

PE can be seen as a generalization of stochastic neighbor embedding (SNE). SNE corresponds to a special case of PE where the objects and classes are identical sets. In SNE, the class posterior probabilities $p(c_k|\mathbf{x}_n)$ are replaced by the probability $p(\mathbf{x}_m|\mathbf{x}_n)$ of object $\mathbf{x}_n$ under a Gaussian distribution centered on $\mathbf{x}_m$. When the inputs (posterior probabilities) to PE come from an unsupervised mixture model, PE performs unsupervised dimensionality reduction just like SNE. However, it has several advantages over SNE and other methods for embedding a single set of data points based on their pairwise relations (e.g., MDS, Isomap). It can be applied in supervised or semi-supervised modes, when class labels are available. Because its computational complexity scales with $NK$, the product of the number of objects and the number of classes, it can be applied efficiently to data sets with very many objects (as long as the number of classes remains small). In this sense, PE is closely related to landmark MDS (LMDS) [2], if we equate classes with landmarks, objects with data points, and $-\log p(c_k|\mathbf{x}_n)$ with the squared distances input to LMDS. However, LMDS lacks a probabilistic semantics and is only suitable for unsupervised settings. Lastly, even if hard classifications are not available, it is often the relations of the objects to the classes, rather than to each other, that we are interested in.

After describing the mathematical formulation and optimization procedures used in PE (Section 2), we present applications to visualizing the structure of several kinds of class posteriors. In section 3, we look at supervised classifiers of hand-labeled web pages. In section 4, we examine semi-supervised classifiers of handwritten digits. Lastly, in section 5, we apply PE to an unsupervised probabilistic topics model, treating latent topics as classes, and words as objects. PE handles these datasets easily, in the last producing an embedding for over 26,000 objects in a little over a minute (on a 2GHz Pentium computer).

## 2   Parametric Embedding method

Given as input conditional probabilities $p(c_k|\mathbf{x}_n)$, PE seeks an embedding of objects with coordinates $\mathbf{r}_n$ and classes with coordinates $\boldsymbol{\phi}_k$, such that $p(c_k|\mathbf{x}_n)$ is approximated as closely as possible by the posterior probabilities from a unit-variance spherical Gaussian mixture model in the embedding space:

$$p(c_k|\mathbf{r}_n) = \frac{p(c_k)\exp(-\frac{1}{2}\parallel \mathbf{r}_n - \boldsymbol{\phi}_k \parallel^2)}{\sum_{l=1}^{K} p(c_l)\exp(-\frac{1}{2}\parallel \mathbf{r}_n - \boldsymbol{\phi}_l \parallel^2)}. \tag{1}$$

Here $\parallel \cdot \parallel$ is the Euclidean norm in the embedding space. When the conditional probabilities $p(c_k|\mathbf{x}_n)$ arise as posterior probabilities from a mixture model, we will also typically be given priors $p(c_k)$ as input; otherwise the $p(c_k)$ terms above may be assumed equal.

It is natural to measure the degree of correspondence between input probabilities and embedding-space probabilities using a sum of KL divergences for each object: $\sum_{n=1}^{N} \mathrm{KL}(p(c_k|\mathbf{x}_n)||p(c_k|\mathbf{r}_n))$. Minimizing this sum w.r.t. $\{p(c_k|\mathbf{r}_n))\}$ is equivalent to minimizing the objective function

$$E(\{\mathbf{r}_n\}, \{\boldsymbol{\phi}_k\}) = -\sum_{n=1}^{N}\sum_{k=1}^{K} p(c_k|\mathbf{x}_n)\log p(c_k|\mathbf{r}_n). \tag{2}$$

Since this minimization problem cannot be solved analytically, we employ a coordinate descent method. We initialize $\{\boldsymbol{\phi}_k\}$, and we iteratively minimize $E$ w.r.t. to $\{\boldsymbol{\phi}_k\}$ or $\{\mathbf{r}_n\}$ while fixing the other set of parameters, until $E$ converges.

Derivatives of $E$ are:

$$\frac{\partial E}{\partial \mathbf{r}_n} = \sum_{k=1}^{K} \alpha_{n,k}(\mathbf{r}_n - \boldsymbol{\phi}_k) \;\; \text{and} \;\; \frac{\partial E}{\partial \boldsymbol{\phi}_k} = \sum_{n=1}^{N} \alpha_{n,k}(\boldsymbol{\phi}_k - \mathbf{r}_n), \tag{3}$$

where $\alpha_{n,k} = p(c_k|\mathbf{x}_n) - p(c_k|\mathbf{r}_n)$. These learning rules have an intuitive interpretation (analogous to those in SNE) as a sum of forces pulling or pushing $\mathbf{r}_n$ ($\boldsymbol{\phi}_k$) depending on the sign of $\alpha_{n,k}$. Importantly, the Hessian of $E$ w.r.t. $\{\mathbf{r}_n\}$ is a semi-positive definite matrix:

$$\frac{\partial^2 E}{\partial \mathbf{r}_n \partial \mathbf{r}'_n} = \sum_{k=1}^{K} p(c_k|\mathbf{r}_n)\boldsymbol{\phi}_k\boldsymbol{\phi}'_k - \left(\sum_{k=1}^{K} p(c_k|\mathbf{r}_n)\boldsymbol{\phi}_k\right)\left(\sum_{k=1}^{K} p(c_k|\mathbf{r}_n)\boldsymbol{\phi}_k\right)' \tag{4}$$

since the r.h.s. of (4) is exactly a covariance matrix. Thus we can find the *globally* optimal solution for $\{\mathbf{r}_n\}$ given $\{\boldsymbol{\phi}_k\}$.[1] The computational complexity of PE is $O(NK)$, which is much more efficient than that of pairwise (dis)similarity-based methods with $O(N^2)$ computations (such as SNE, MDS, or Isomap).

## 3 Analyzing supervised classifiers on web data

In this section, we show how PE can be used to visualize the structure of labeled data (web pages) in a supervised classification task. We also compare PE with two conventional methods, MDS [7] and Fisher linear discriminant analysis (FLDA) [3]. MDS seeks a low-dimensional embedding that preserves the input distances between objects. It does not normally use class labels for data points, although below we discuss a way to apply MDS to label probabilities that arise in classification. FLDA, in contrast, naturally uses labeled data in constructing a low-dimensional embedding. It seeks a a linear projection of the objects' coordinates in a high-dimensional ambient space that maximizes between-class variance and minimizes within-class variance.

The set of objects comprised 5500 human-classified web pages: 500 pages sampled from each of 11 top level classes in Japanese directories of Open Directory (http://dmoz.org/). Pages with less than 50 words, or which occurred under multiple categories, were eliminated. A Naive Bayes (NB) classifier was trained on the full data (represented as word frequency vectors). Posterior probabilities $p(c_k|\mathbf{x}_n)$ were calculated for classifying each object (web page), assuming its true class label was unknown. These probabilities, as well as estimated priors $p(c_k)$, form the input to PE.

Fig.1(a) shows the output of PE, which captures many features of this data set and classification algorithm. Pages belonging to the same class tend to cluster well in the embedding, which makes sense given the large sample of labeled data. Related categories are located nearby: e.g., sports and health, or computers and online-shopping. Well-separated clusters correspond to classes (e.g. sports) that are easily distinguished from others. Conversely, regional pages are dispersed, indicating that they are not easily classified. Distinctive pages are evident as well: a few pages that are scattered among the objects of another category might be misclassified. Pages located between clusters are likely to be categorized in multiple classes; arcs between two classes show subsets of objects that distribute their probability among those two classes and no others.

Fig.1(b) shows the result of MDS applied to cosine distances between web pages. No labeled information is used (only word frequency vectors for the pages), and consequently no class structure is visible. Fig.1(c) shows the result of FLDA. To stabilize the calculation, FLDA was applied only after word frequencies were smoothed via SVD. FLDA uses label information, and clusters together the objects in each class better than MDS does. However, most clusters are highly overlapping, and the separation of classes is much poorer than with PE. This seems to be a consequence of FLDA's restriction to purely linear projections, which cannot, in general, separate all of the classes.

Fig.1(d) shows another way of embedding the data using MDS, but this time applied to Euclidean distances in the $(K-1)-$dimensional space of posterior distributions $p(c_k|\mathbf{x}_n)$. Pages belonging to the same class are definitely more clustered in this mode, but still the clusters are highly overlapping and provide little insight into the classifier's behavior. This version of MDS uses the same inputs as PE, rather than any high-dimensional word frequency vectors, but its computations are not explicitly probabilistic. The superior results of PE (Fig.1(a)) illustrate the advantage of optimizing an appropriate probabilistic objective function.

## 4 Application to semi-supervised classification

The utility of PE for analyzing classifier performance may best be illustrated in a semi-supervised setting, with a large unlabeled set of objects and a smaller set of labeled objects. We fit a probabilistic classifier based on the labeled objects, and we would like to visualize the behavior of the classifier applied to the unlabeled objects, in a way that suggests how

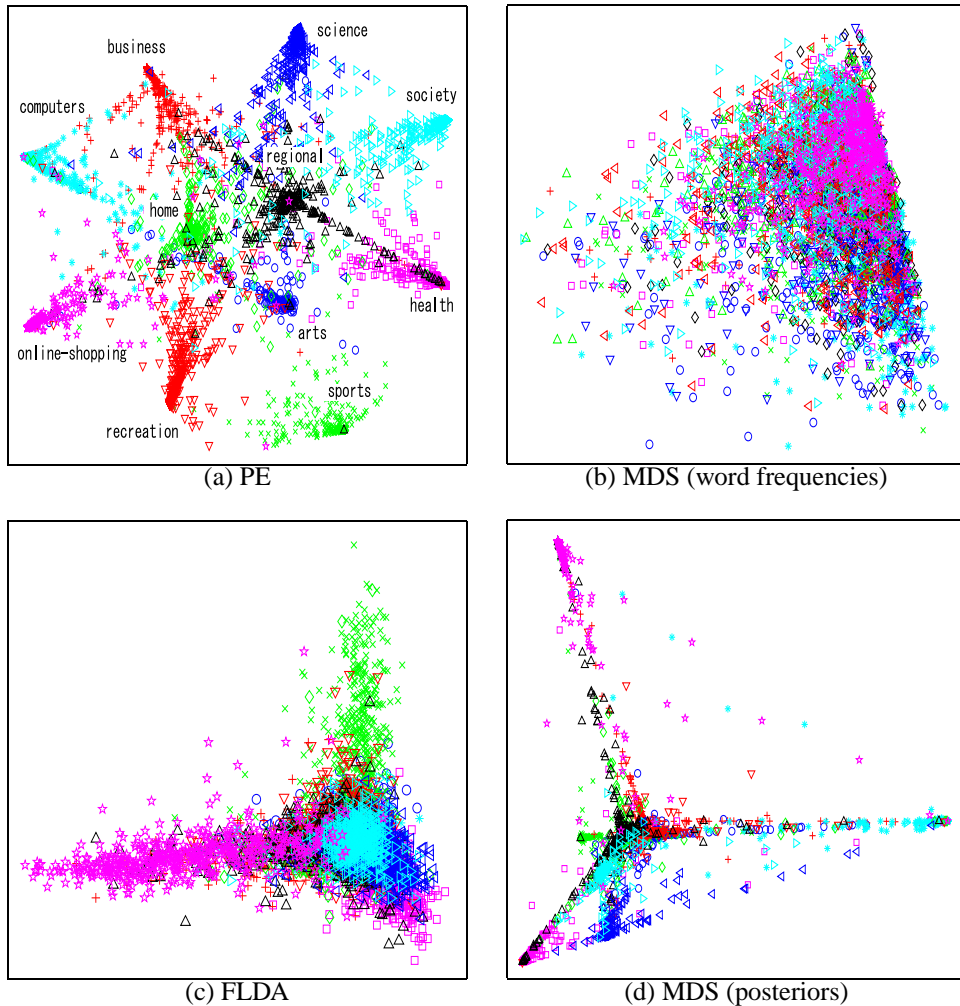

Figure 1: The visualizations of categorized web pages. Each of the 5500 web pages is show by a particle with shape indicating the page's class.

accurate the classifier is likely to be and what kinds of errors it is likely to make.

We constructed a simple probabilistic classifier for 2558 handwritten digits (classes 0-4) from the MNIST database. The classifier was based on a mixture model for the density of each class, defined by selecting either 10 or 100 digits uniformly at random from each class and centering a fixed-covariance Gaussian (in pixel space) on each of these examples – essentially a soft nearest-neighbor method. The posterior distribution over this classifier for all 2558 digits was submitted as input to PE.

The resulting embeddings allow us to predict the classifiers' patterns of confusions, calculated based on the true labels for all 2558 objects. Fig. 2 shows embeddings for both 10 labels/class and 100 labels/class. In both cases we see five clouds of points corresponding to the five classes. The clouds are elongated and oriented roughly towards a common center, forming a star shape (also seen to some extent in our other applications). Objects that concentrate their probability on only one class will lie as far from the center of the plot as possible – ideally, even farther than the mean of their class, because this maximizes their

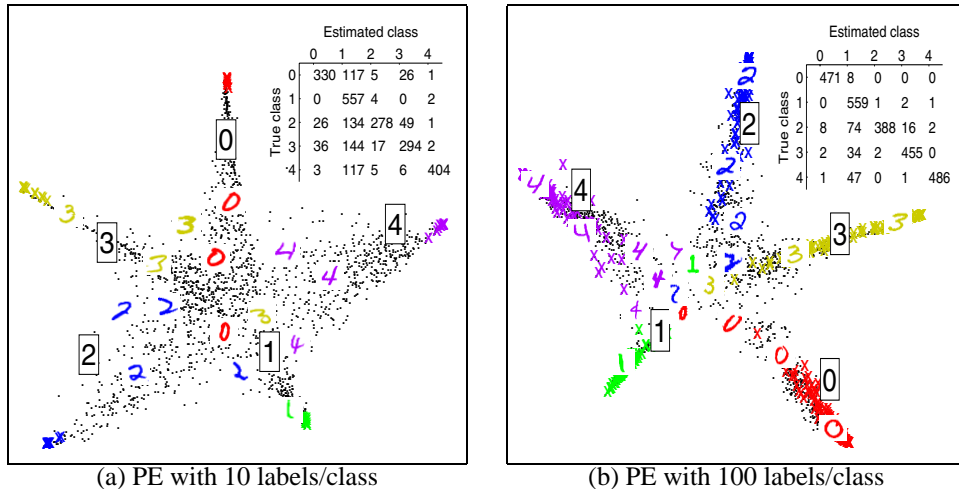

(a) PE with 10 labels/class     (b) PE with 100 labels/class

Figure 2: Parametric embeddings for handwritten digit classification. Each dot represents the coordinates $\mathbf{r}_n$ of one image. Boxed numbers represent the class means $\phi_k$. ×'s show labeled examples used to train the classifier. Images of several unlabeled digits are shown for each class.

posterior probability on that class. Moving towards the center of the plot, objects become increasingly confused with other classes.

Relative to using only 10 labels/class, using 100 labels yields clusters that are more distinct, reflecting better between-class discrimination. Also, the labeled examples are more evenly spread through each cluster, reflecting more faithful within-class models and less overfitting. In both cases, the '1' class is much closer than any other to the center of the plot, reflecting the fact that instances of other classes tend to be mistaken for '1's. Instances of other classes near the '1' center also tend to look rather "one-like" – thinner and more elongated. The dense cluster of points just outside the mean for '1' reflects the fact that '1's are rarely mistaken for other digits. In Fig. 2(a), the '0' and '3' distributions are particularly overlapping, reflecting that those two digits are most readily confused with each other (apart from 1). The 'webbing' between the diffuse '2' arm and the tighter '3' arm reflects the large number of '2's taken for '3's. In Fig. 2(b), that 'webbing' persists, consistent with the observation that (again, apart from many mistaken responses of 1) the confusion of '2's for '3's is the only large-scale error these larger data permit.

## 5   Application to unsupervised latent class models

In the applications above, PE was applied to visualize the structure of classes based at least to some degree on labeled examples. The algorithm can also be used in a completely unsupervised setting, to visualize the structure of a probabilistic generative model based on latent classes. Here we illustrate this application of PE by visualizing a semantic space of word meanings: objects correspond to words, and classes correspond to topics in a latent Dirichlet allocation (LDA) model [1] fit to a large (>37,000 documents, >12,000,000 word tokens) corpus of educational materials for first grade to college (TASA). The problem of mapping a large vocabulary is particularly challenging, and, with over 26,000 objects (word types), prohibitively expensive for pairwise methods. Again, PE solves for the configuration shown in about a minute.

In LDA (not to be confused with FLDA above), each topic defines a probability distribution

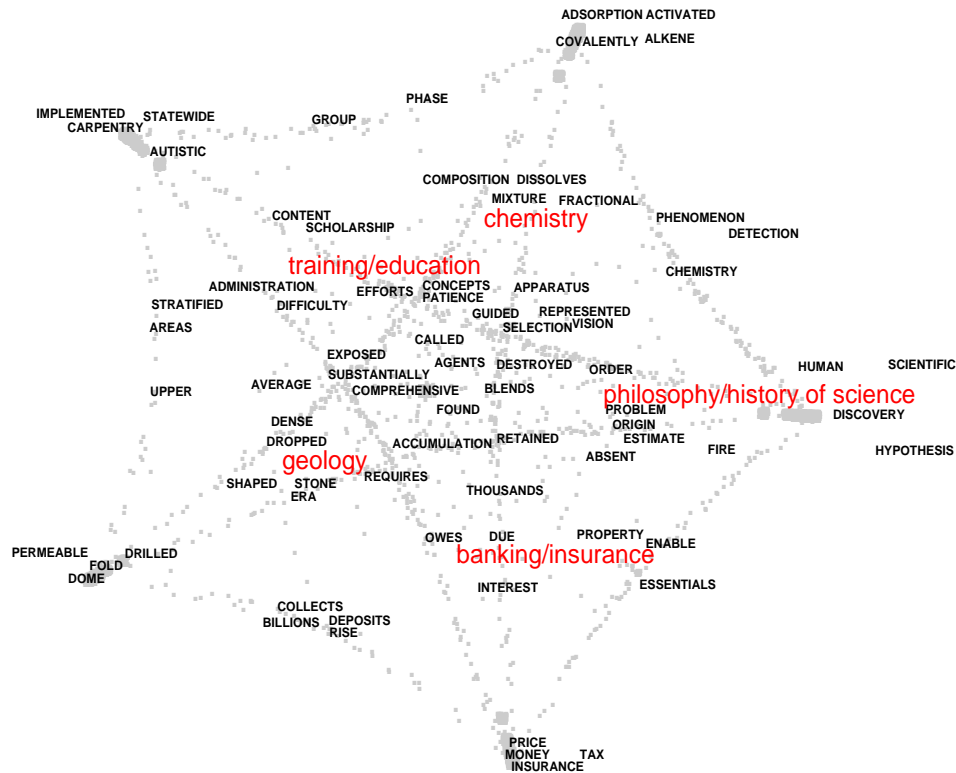

Figure 3: Parametric embedding for word meanings and topics based on posterior distributions from an LDA model. Each dot represents the coordinates $\mathbf{r}_n$ of one word. Large phrases indicate the positions of topic means $\phi_k$ (with topics labeled intuitively). Examples of words that belong to one or more topics are also shown.

over word types that can occur in a document. This model can be inverted to give the probability that topic $c_k$ was responsible for generating word $x_n$; these probabilities $p(c_k|\mathbf{x}_n)$ provide the input needed to construct a space of word and topic meanings in PE.

More specifically, we fit a 50-topic LDA model to the TASA corpus. Then, for each word type, we computed its posterior distribution restricted to a subset of 5 topics, and input these conditional probabilities to PE (with $N = 26,243$, $K = 5$). Fig. 3 shows the resulting embedding. As with the embeddings in Figs. 1 and 2, the topics are arranged roughly in a star shape, with a tight cluster of points at each corner of the star corresponding to words that place almost all of their probability mass on that topic. Semantically, the words in these extreme clusters often (though not always) have a fairly specialized meaning particular to the nearest topic. Moving towards the center of the plot, words take on increasingly general meanings.

This embedding shows other structures not visible in previous figures: in particular, dense curves of points connecting every pair of clusters. This pattern reflects the characteristic probabilistic structure of topic models of semantics: in addition to the clusters of words that associate with just one topic, there are many words that associate with just two topics, or just three, and so on. The dense curves in Fig. 3 show that for any pair of topics in this corpus, there exists a substantial subset of words that associate with just those topics.

For words with probability sharply concentrated on two topics, points along these curves minimize the sum of the KL and regularization terms. This kind of distribution is intrinsically high-dimensional and cannot be captured with complete fidelity in any 2-dimensional embedding.

As shown by the examples labeled in Fig. 3, points along the curves connecting two apparently unrelated topics often have multiple meanings or senses that join them to each topic: 'deposit' has both a geological and a financial sense, 'phase' has both an everyday and a chemical sense, and so on.

## 6   Conclusions

We have proposed a probabilistic embedding method, PE, that embeds objects and classes simultaneously. PE takes as input a probability distribution for objects over classes, or more generally of one set of points over another set, and attempts to fit that distribution with a simple class-conditional parametric mixture in the embedding space. Computationally, PE is inexpensive relative to methods based on similarities or distances between all pairs of objects, and converges quickly on many thousands of data points.

The visualization results of PE shed light on features of both the data set and the classification model used to generate the input conditional probabilities, as shown in applications to classified web pages, partially classified digits, and the latent topics discovered by an unsupervised method, LDA. PE may also prove useful for similarity-preserving dimension reduction, where the high-dimensional model is not of primary interest, or more generally, in analysis of large conditional probability tables that arise in a range of applied domains.

As an example of an application we have not yet explored, purchases, web-surfing histories, and other preference data naturally form distributions over items or categories of items. Conversely, items define distributions over people or categories thereof. Instances of such dyadic data abound–restaurants and patrons, readers and books, authors and publications, species and foods...–with patterns that might be visualized. PE provides a tractable, principled, and effective visualization method for large volumes of such data, for which pairwise methods are not appropriate.

**Acknowledgments**

This work was supported by a grant from the NTT Communication Sciences Laboratories.

## Footnotes

[1] In our experiments, we found that optimization proceeded more smoothly with a regularized objective function, $J = E + \eta_r \sum_{n=1}^{N} \parallel \mathbf{r}_n \parallel^2 + \eta_\phi \sum_{k=1}^{K} \parallel \boldsymbol{\phi}_k \parallel^2$, where $\eta_r, \eta_\phi > 0$.

## References

[1]  D. Blei, A. Ng and M. Jordan. Latent dirichlet allocation. *NIPS* 15, 2002.

[2]  V. de Silva, J. B. Tenenbaum. Global versus local methods in nonlinear dimensionality reduction. *NIPS* 15, pp. 705-712, 2002.

[3]  R. Fisher. The use of multiple measurements in taxonomic problem. *Annuals of Eugenics* 7, pp.179–188, 1950.

[4]  G. Hinton and S. Roweis. Stochastic neighbor embedding. *NIPS* 15, 2002.

[5]  I.T. Joliffe. *Principal Component Analysis*. Springer, 1980.

[6]  J. Tenenbaum, V. de Silva and J. Langford. A global geometric framework for nonlinear dimensionality reduction. *Science* 290 pp. 2319–2323, 2000.

[7]  W. Torgerson. *Theory and Methods of Scaling*. New York, Wiley, 1958.
